# Label Ranking with Partial Abstention based on Thresholded Probabilistic Models

**Weiwei Cheng**
Mathematics and Computer Science
Philipps-Universität Marburg
Marburg, Germany
cheng@mathematik.uni-marburg.de

**Eyke Hüllermeier**
Mathematics and Computer Science
Philipps-Universität Marburg
Marburg, Germany
eyke@mathematik.uni-marburg.de

**Willem Waegeman**
Mathematical Modeling, Statistics and
Bioinformatics, Ghent University
Ghent, Belgium
willem.waegeman@ugent.be

**Volkmar Welker**
Mathematics and Computer Science
Philipps-Universität Marburg
Marburg, Germany
welker@mathematik.uni-marburg.de

## Abstract

Several machine learning methods allow for abstaining from uncertain predictions. While being common for settings like conventional classification, abstention has been studied much less in learning to rank. We address abstention for the label ranking setting, allowing the learner to declare certain pairs of labels as being incomparable and, thus, to predict partial instead of total orders. In our method, such predictions are produced via thresholding the probabilities of pairwise preferences between labels, as induced by a predicted probability distribution on the set of all rankings. We formally analyze this approach for the Mallows and the Plackett-Luce model, showing that it produces proper partial orders as predictions and characterizing the expressiveness of the induced class of partial orders. These theoretical results are complemented by experiments demonstrating the practical usefulness of the approach.

## 1 Introduction

In machine learning, the notion of "abstention" commonly refers to the possibility of refusing a prediction in cases of uncertainty. In classification with a *reject option*, for example, a classifier may abstain from a class prediction if making no decision is considered less harmful than making an unreliable and hence potentially false decision [7, 1]. The same idea could be used in the context of *ranking*, too, where a reject option appears to be even more interesting than in classification. While a conventional classifier has only two choices, namely to predict a class or to abstain, a ranker can abstain to some degree: The order relation predicted can be more or less complete, ranging from a total order to the empty relation in which all alternatives are declared incomparable.

Our focus is on so-called *label ranking* problems [16, 10], to be introduced more formally in Section 2 below. Label ranking has a strong relationship with the standard setting of multi-class classification, but each instance is now associated with a complete ranking of all labels instead of a single label. Typical examples, which also highlight the need for abstention, include the ranking of candidates for a given job and the ranking of products for a given customer. In such applications, it is desirable to avoid the expression of unreliable or unwarranted preferences. Thus, if a ranker cannot reliably decide whether a first label should precede a second one or the other way around, it should abstain from this decision and instead declare these alternatives as being incomparable. Abstaining in a consistent way, the relation thus produced should form a partial order [6].

In Section 4, we propose and analyze a new approach for abstention in label ranking that builds on existing work on partial orders in areas like decision theory, probability theory and discrete mathematics. We predict partial orders by thresholding parameterized probability distributions on rankings, using the Plackett-Luce and the Mallows model. Roughly speaking, this approach is able to avoid certain inconsistencies of a previous approach to label ranking with abstention [6], to be discussed in Section 3. By making stronger model assumptions, our approach simplifies the construction of consistent partial order relations. In fact, it obeys a number of appealing theoretical properties. Apart from assuring proper partial orders as predictions, it allows for an exact characterization of the expressivity of a class of thresholded probability distributions in terms of the number of partial orders that can be produced. The proposal and formal analysis of this approach constitute our main contributions.

Last but not least, as will be shown in Section 5, the theoretical advantages of our approach in comparison with [6] are also reflected in practical improvements.

## 2 Label Ranking with Abstention

In the setting of label ranking, each instance $x$ from an instance space $\mathbb{X}$ is associated with a total order of a fixed set of class labels $\mathcal{Y} = \{y_1, \ldots, y_M\}$, that is, a complete, transitive, and antisymmetric relation $\succ$ on $\mathcal{Y}$, where $y_i \succ y_j$ indicates that $y_i$ precedes $y_j$ in the order. Since a ranking can be considered as a special type of preference relation, we shall also say that $y_i \succ y_j$ indicates a preference for $y_i$ over $y_j$ (given the instance $x$).

Formally, a total order $\succ$ can be identified with a permutation $\pi$ of the set $[M] = \{1, \ldots, M\}$, such that $\pi(i)$ is the position of $y_i$ in the order. We denote the class of permutations of $[M]$ (the symmetric group of order $M$) by $\Omega$. Moreover, we identify $\succ$ with the mapping (relation) $R : \mathcal{Y}^2 \longrightarrow \{0, 1\}$ such that $R(i, j) = 1$ if $y_i \succ y_j$ and $R(i, j) = 0$ otherwise.

The goal in label ranking is to learn a "label ranker" in the form of an $\mathbb{X} \longrightarrow \Omega$ mapping. As training data, a label ranker uses a set of instances $x_n$ ($n \in [N]$), together with preference information in the form of pairwise comparisons $y_i \succ y_j$ of some (but not necessarily all) labels in $\mathcal{Y}$, suggesting that instance $x_n$ prefers label $y_i$ to $y_j$.

The prediction accuracy of a label ranker is assessed by comparing the true ranking $\pi$ with the prediction $\hat{\pi}$, using a distance measure $D$ on rankings. Among the most commonly used measures is the Kendall distance, which is defined by the number of inversions, that is, pairs $\{i, j\} \subset [M]$ such that $\text{sign}(\pi(i) - \pi(j)) \neq \text{sign}(\hat{\pi}(i) - \hat{\pi}(j))$. Besides, the sum of squared rank distances, $\sum_{i=1}^{M} (\pi(i) - \hat{\pi}(i))^2$, is often used as an alternative distance; it is closely connected to Spearman's rank correlation (Spearman's rho), which is an affine transformation of this number to the interval $[-1, +1]$.

Motivated by the idea of a reject option in classification, Cheng et al. [6] introduced a variant of the above setting in which the label ranker is allowed to partially abstain from a prediction. More specifically, it is allowed to make predictions in the form of a *partial* order $Q$ instead of a *total* order $R$: If $Q(i, j) = Q(j, i) = 0$, the ranker abstains on the label pair $(y_i, y_j)$ and instead declares these labels as being incomparable. Abstaining in a consistent way, $Q$ should still be antisymmetric and transitive, hence a partial order relation. Note that a prediction $Q$ can be associated with a *confidence set*, i.e., a subset of $\Omega$ supposed to cover the true ranking $\pi$, namely the set of all linear extensions of this partial order: $\mathcal{C}(Q) = \{\pi \in \Omega \mid Q(i, j) = 1 \Rightarrow (\pi(i) < \pi(j)) \text{ for all } i, j \in [M]\}$.

## 3 Previous Work

Despite a considerable amount of work on ranking in general and learning to rank in particular, the literature on *partial* rankings is relatively sparse. Worth mentioning is work on a specific type of partial orders, namely linear orders of unsorted or tied subsets (partitions, bucket orders) [13, 17]. However, apart from the restriction to this type of order relation, the problems addressed in these works are quite different from our goals. The authors in [17] specifically address computational aspects that arise when working with distributions on partially ranked data, while [13] seeks to discover an underlying bucket order from pairwise precedence information between the items.

More concretely, in the context of the label ranking problem, the aforementioned work [6] is the only one so far that addresses the more general problem of learning to predict partial orders. This method consists of two main steps and can be considered as a *pairwise* approach in the sense that, as a point of departure for a prediction, a valued preference relation $P : \mathcal{Y}^2 \longrightarrow [0, 1]$ is produced, where $P(i, j)$ is interpreted as a measure of support of the pairwise preference $y_i \succ y_j$. Support is commonly interpreted in terms of probability, hence $P$ is assumed to be reciprocal: $P(i, j) = 1 - P(j, i)$ for all $i, j \in [M]$.

Then, in a second step, a partial order $Q$ is derived from $P$ via thresholding:

$$Q(i, j) = [\![ P(i, j) > q ]\!] = \begin{cases} 1, & \text{if } P(i, j) > q \\ 0, & \text{otherwise} \end{cases}, \tag{1}$$

where $1/2 \le q < 1$ is a threshold. Thus, the idea is to predict only those pairwise preferences that are sufficiently likely, while abstaining on pairs $(i, j)$ for which $P(i, j) \approx 1/2$.

The first step of deriving the relation $P$ is realized by means of an ensemble learning technique: Training an ensemble of standard label rankers, each of which provides a prediction in the form of a total order, $P(i, j)$ is defined by the fraction of ensemble members voting for $y_i \succ y_j$. Other possibilities are of course conceivable, and indeed, the only important point to notice here is that the preference degrees $P(i, j)$ are essentially independent of each other. Or, stated differently, they do not guarantee any specific properties of the relation $P$ except being reciprocal. In particular, $P$ does not necessarily obey any type of transitivity property.

For the relation $Q$ derived from $P$ via thresholding, this has two important consequences: First, if the threshold $q$ is not large enough, then $Q$ may have cycles. Thus, not all thresholds in $[1/2, 1)$ are actually feasible. In particular, if $q = 1/2$ cannot be chosen, this also implies that the method may not be able to predict a total order as a special case. Second, even if $Q$ does not have cycles, it is not guaranteed to be transitive.

To overcome these problems, the authors devise an algorithm (of complexity $\mathcal{O}(|\mathcal{Y}|^3)$) that finds the smallest feasible threshold $q_{min}$, namely the threshold that guarantees $Q(i, j) = [\![ P(i, j) > q ]\!]$ to be cycle-free for each threshold $q \in [q_{min}, 1)$. Then, since $Q$ may still be non-transitive, it is "repaired" in a second step by replacing it with its transitive closure [23].

## 4 Predicting Partial Orders based on Probabilistic Models

In order to tackle the problems of the approach in [6], our idea is to restrict the relation $P$ in (1) so as to exclude the possibility of cycles and intransitivity from the very beginning, thereby avoiding the need for a post-reparation of a prediction. To this end, we take advantage of methods for label ranking that produce (parameterized) *probability distributions* over $\Omega$ as predictions. Our main theoretical result is to show that thresholding pairwise preferences induced by such distributions, apart from being semantically meaningful due to their clear probabilistic interpretation, yields preference relations with the desired properties, that is, partial order relations $Q$.

### 4.1 Probabilistic Models

Different types of probability models for rank data have been studied in the statistical literature [11, 20], including the Mallows model and the Plackett-Luce (PL) model as the most popular representatives of the class of *distance-based* and *stagewise* models, respectively. Both models have recently attracted attention in machine learning [14, 15, 22, 21, 18] and, in particular, have been used in the context of label ranking.

A label ranking method that produces predictions expressed in terms of the Mallows model is proposed in [5]. The standard Mallows model

$$\mathbf{P}(\pi \,|\, \theta, \pi_0) = \frac{\exp(-\theta D(\pi, \pi_0))}{\phi(\theta)} \tag{2}$$

is determined by two parameters: The ranking $\pi_0 \in \Omega$ is the location parameter (mode, center ranking) and $\theta \ge 0$ is a spread parameter. Moreover, $D$ is a distance measure on rankings, and $\phi = \phi(\theta)$ is a normalization factor that depends on the spread (but, provided the right-invariance

of $D$, not on $\pi_0$). Obviously, the Mallows model assigns the maximum probability to the center ranking $\pi_0$. The larger the distance $D(\pi, \pi_0)$, the smaller the probability of $\pi$ becomes. The spread parameter $\theta$ determines how quickly the probability decreases, i.e., how peaked the distribution is around $\pi_0$. For $\theta = 0$, the uniform distribution is obtained, while for $\theta \to \infty$, the distribution converges to the one-point distribution that assigns probability 1 to $\pi_0$ and 0 to all other rankings.

Alternatively, the Plackett-Luce (PL) model was used in [4]. This model is specified by a parameter vector $\boldsymbol{v} = (v_1, v_2, \ldots, v_M) \in \mathbb{R}_+^M$:

$$\mathbf{P}(\pi \mid \boldsymbol{v}) = \prod_{i=1}^{M} \frac{v_{\pi^{-1}(i)}}{v_{\pi^{-1}(i)} + v_{\pi^{-1}(i+1)} + \ldots + v_{\pi^{-1}(M)}} \tag{3}$$

It is a generalization of the well-known Bradley-Terry model for the pairwise comparison of alternatives, which specifies the probability that "$a$ wins against $b$" in terms of $\mathbf{P}(a \succ b) = \frac{v_a}{v_a + v_b}$. Obviously, the larger $v_a$ in comparison to $v_b$, the higher the probability that $a$ is chosen. Likewise, the larger the parameter $v_i$ in (3) in comparison to the parameters $v_j$, $j \neq i$, the higher the probability that $y_i$ appears on a top rank.

## 4.2 Thresholded Relations are Partial Orders

Given a probability distribution $\mathbf{P}$ on the set of rankings $\Omega$, the probability of a pairwise preference $y_i \succ y_j$ (and hence the corresponding entry in the preference relation $P$) is obtained through marginalization:

$$P(i,j) = \mathbf{P}(y_i \succ y_j) = \sum_{\pi \in E(i,j)} \mathbf{P}(\pi) \ , \tag{4}$$

where $E(i,j)$ denotes the set of linear extensions of the incomplete ranking $y_i \succ y_j$, i.e., the set of all rankings $\pi \in \Omega$ with $\pi(i) < \pi(j)$. We start by stating a necessary and sufficient condition on $P(i,j)$ for the threshold relation (1) to result in a (strict) partial order, i.e., an antisymmetric, irreflexive and transitive relation.

**Lemma 1.** *Let $P$ be a reciprocal relation and let $Q$ be given by (1). Then $Q$ defines a strict partial order relation for all $q \in [1/2, 1)$ if and only if $P$ satisfies partial stochastic transitivity, i.e., $P(i,j) > 1/2$ and $P(j,k) > 1/2$ implies $P(i,k) \geq \min(P(i,j), P(j,k))$ for each triple $(i,j,k) \in [M]^3$.*

This lemma was first proven by Fishburn [12], together with a number of other characterizations of subclasses of strict partial orders by means of transitivity properties on $P(i,j)$. For example, replacing partial stochastic transitivity by interval stochastic transitivity (now a condition on quadruples instead of triplets) leads to a characterization of interval orders, a subclass of strict partial orders; a partial order $Q$ on $[M]^2$ is called an interval order if each $i \in [M]$ can be associated with an interval $(l_i, u_i) \subset \mathbb{R}$ such that $Q(i,j) = 1 \Leftrightarrow u_i \leq l_j$.

Our main theoretical results below state that thresholding (4) yields a strict partial order relation $Q$, both for the PL and the Mallows model. Thus, we can guarantee that a strict partial order relation can be predicted by simple thresholding, and without the need for any further reparation. Moreover, the whole spectrum of threshold parameters $q \in [1/2, 1)$ can be used.

**Theorem 1.** *Let $\mathbf{P}$ in (4) be the PL model (3). Moreover, let $Q$ be given by the threshold relation (1). Then $Q$ defines a strict partial order relation for all $q \in [1/2, 1)$.*

**Theorem 2.** *Let $\mathbf{P}$ in (4) be the Mallows model (2), with a distance $D$ having the so-called transposition property. Moreover, let $Q$ be given by the threshold relation (1). Then $Q$ defines a strict partial order relation for all $q \in [1/2, 1)$.*

Theorem 1 directly follows from the strong stochastic transitivity of the PL model [19]. The proof of Theorem 2 is slightly more complicated and given below. Moreover, the result for Mallows is less general in the sense that $D$ must obey the *transposition property*. Actually, however, this property is not very restrictive and indeed satisfied by most of the commonly used distance measures, including the Kendall distance (see, e.g., [9]). In the following, we always assume that the distance $D$ in the Mallows model (2) satisfies this property.

**Definition 1.** *A distance $D$ on $\Omega$ is said to have the transposition property, if the following holds: Let $\pi$ and $\pi'$ be rankings and let $(i,j)$ be an inversion (i.e., $i < j$ and $(\pi(i) - \pi(j))(\pi'(i) - \pi'(j)) < 0$). Let $\pi'' \in \Omega$ be constructed from $\pi'$ by swapping $y_i$ and $y_j$, that is, $\pi''(i) = \pi'(j)$, $\pi''(j) = \pi'(i)$ and $\pi''(m) = \pi'(m)$ for all $m \in [M] \setminus \{i,j\}$. Then, $D(\pi, \pi'') \leq D(\pi, \pi')$.*

**Lemma 2.** *If $y_i$ precedes $y_j$ in the center ranking $\pi_0$ in (2), then $\mathbf{P}(y_i \succ y_j) \geq 1/2$. Moreover, if $\mathbf{P}(y_i \succ y_j) > q \geq 1/2$, then $y_i$ precedes $y_j$ in the center ranking $\pi_0$.*

*Proof.* For every ranking $\pi \in \Omega$, let $b(\pi) = \pi$ if $y_i$ precedes $y_j$ in $\pi$; otherwise, $b(\pi)$ is defined by swapping $y_i$ and $y_j$ in $\pi$. Obviously, $b(\cdot)$ defines a bijection between $E(i,j)$ and $E(j,i)$. Moreover, since $D$ has the transposition property, $D(b(\pi), \pi_0) \leq D(\pi, \pi_0)$ for all $\pi \in \Omega$. Therefore, according to the Mallows model, $\mathbf{P}(b(\pi)) \geq \mathbf{P}(\pi)$, and hence

$$\mathbf{P}(y_i \succ y_j) = \sum_{\pi \in E(i,j)} \mathbf{P}(\pi) \geq \sum_{\pi \in E(i,j)} \mathbf{P}(b^{-1}(\pi)) = \sum_{\pi \in E(j,i)} \mathbf{P}(\pi) = \mathbf{P}(y_j \succ y_i)$$

Since, moreover, $\mathbf{P}(y_i \succ y_j) = 1 - \mathbf{P}(y_j \succ y_i)$, it follows that $\mathbf{P}(y_i \succ y_j) \geq 1/2$. The second part immediately follows from the first one by way of contradiction: If $y_j$ would precede $y_i$, then $\mathbf{P}(y_j \succ y_i) \geq 1/2$, and therefore $\mathbf{P}(y_i \succ y_j) = 1 - \mathbf{P}(y_j \succ y_i) \leq 1/2 \leq q$. $\square$

**Lemma 3.** *If $y_i$ precedes $y_j$ and $y_j$ precedes $y_k$ in the center ranking $\pi_0$ in (2), then $\mathbf{P}(y_i \succ y_k) \geq \max\left(\mathbf{P}(y_i \succ y_j), \mathbf{P}(y_j \succ y_k)\right)$.*

*Proof.* We show that $\mathbf{P}(y_i \succ y_k) \geq \mathbf{P}(y_i \succ y_j)$. The second inequality $\mathbf{P}(y_i \succ y_k) \geq \mathbf{P}(y_j \succ y_k)$ is shown analogously. Let $E(i,j,k)$ denote the set of linear extensions of $y_i \succ y_j \succ y_k$, i.e., the set of rankings $\pi \in \Omega$ in which $y_i$ precedes $y_j$ and $y_j$ precedes $y_k$. Now, for every $\pi \in E(k,j,i)$, define $b(\pi)$ by first swapping $y_k$ and $y_j$ and then $y_k$ and $y_i$ in $\pi$. Obviously, $b(\cdot)$ defines a bijection between $E(k,j,i)$ and $E(j,i,k)$. Moreover, due to the transposition property, $D(b(\pi), \pi_0) \leq D(\pi, \pi_0)$, and therefore $\mathbf{P}(b(\pi)) \geq \mathbf{P}(\pi)$ under the Mallows model. Consequently, since $E(i,j) = E(i,j,k) \cup E(i,k,j) \cup E(k,i,j)$ and $E(i,k) = E(i,k,j) \cup E(i,j,k) \cup E(j,i,k)$, it follows that $\mathbf{P}(y_i \succ y_k) - \mathbf{P}(y_i \succ y_j) = \sum_{\pi \in E(i,k) \setminus E(i,j)} \mathbf{P}(\pi) = \sum_{\pi \in E(j,i,k)} \mathbf{P}(\pi) - \sum_{\pi \in E(k,j,i)} \mathbf{P}(\pi) = \sum_{\pi \in E(k,j,i)} \mathbf{P}(b(\pi)) - \mathbf{P}(\pi) \geq 0$. $\square$

Lemmas 2 and 3 immediately imply the following lemma.

**Lemma 4.** *The relation $P$ derived via $P(i,j) = \mathbf{P}(y_i \succ y_j)$ from the Mallows model satisfies the following property (closely related to strong stochastic transitivity): If $(P(i,j) > q$ and $P(j,k) > q$, then $P(i,k) \geq \max(P(i,j), P(j,k))$, for all $q \geq 1/2$ and all $i,j,k \in [M]$.*

*Proof of Theorem 2.* Since $\mathbf{P}(y_i \succ y_j) = 1 - \mathbf{P}(y_j \succ y_i)$, it obviously follows that $Q(y_i, y_j) = 1$ implies $Q(y_j, y_i) = 0$. Moreover, Lemma 4 implies that $Q$ is transitive. Consequently, $Q$ defines a proper partial order relation. $\square$

The above statements guarantee that a strict partial order relation can be predicted by simple thresholding, and without the need for any further reparation. Moreover, the whole spectrum of threshold parameters $q \in [1/2, 1)$ can be used. As an aside, we mention that strict partial orders can also be produced by thresholding other probabilistic preference learning models. All pairwise preference models based on utility scores satisfy strong stochastic transitivity. This includes traditional statistical models such as the Thurstone Case 5 model [25] and the Bradley-Terry model [3], as well as modern learning models such as [8, 2]. These models are usually not applied in label ranking settings, however.

### 4.3 Expressivity of the Model Classes

So far, we have shown that predictions produced by thresholding probability distributions on rankings are proper partial orders. Roughly speaking, this is accomplished by restricting $P$ in (1) to specific valued preference relations (namely marginals (4) of the Mallows or the PL model), in contrast to the approach of [6], where $P$ can be any (reciprocal) relation. From a learning point of view, one may wonder to what extent this restriction limits the expressivity of the underlying model class. This expressivity is naturally defined in terms of the number of different partial orders (up to

isomorphism) that can be represented in the form of a threshold relation (1). Interestingly, we can show that, in this sense, the approach based on PL is much more expressive than the one based on the Mallows model.

**Theorem 3.** *Let $\mathcal{Q}_M$ denote the set of different partial orders (up to isomorphism) that can be represented as a threshold relation $Q$ defined by (1), where $P$ is derived according to (4) from the Mallows model (2) with $D$ the Kendall distance. Then $|\mathcal{Q}_M| = M$.*

*Proof.* By the right invariance of $D$, different choices of $\pi_0$ lead to the same set of isomorphism classes $\mathcal{Q}_M$. Hence we may assume that $\pi_0$ is the identity. By Theorem 6.3 in [20] the $(M \times M)$-matrix with entries $P(i, j)$ is a Toeplitz matrix, i.e., $P(i, j) = P(i+1, j+1)$ for all $i, j \in [M-1]$, with entries strictly increasing along rows, i.e., $P(i, j) < P(i, j+1)$ for $1 \le i < j < M$. Thus, by Theorem 2, thresholding leads to $M$ different partial orders. $\square$

More specifically, the partial orders in $\mathcal{Q}_M$ have a very simple structure that is purely rank-dependent: The first structure is the total order $\pi = \pi_0$. The second structure is obtained by removing all preferences between all direct neighbors, i.e., labels $y_i$ and $y_j$ with adjacent ranks ($|\pi(i) - \pi(j)| = 1$). The third structure is obtained from the second one by removing all preferences between 2-neighbors, i.e., labels $y_i$ and $y_j$ with ($|\pi(i) - \pi(j)| = 2$), and so forth.

The cardinality of $\mathcal{Q}_M$ increases for distance measures $D$ other than Kendall (like Spearman's rho or footrule), mainly since in general the matrix with entries $P(i, j)$ is no longer Toeplitz. However, for some measures, including the two just mentioned, the matrix will still be symmetric with respect to the antidiagonal, i.e., $P(i, j) = P(M + 1 - i, M + 1 - j)$ for $j > i$) and have entries increasing along rows. While the exact counting of $\mathcal{Q}_M$ appears to be very difficult in such cases, an argument similar to the one used in the proof of the next result shows that $|\mathcal{Q}_M|$ is bounded by the number of symmetric Dyck paths and hence $|\mathcal{Q}_M| \le \binom{M}{\lfloor \frac{M}{2} \rfloor}$ (see Ch. 7 [24]). It is a simple consequence of Theorem 4 below, showing that exponentially more orders can be produced based on the PL model.

**Lemma 5.** *For fixed $q \in (1/2, 1)$ and a set $A$ of subsets of $[M]$, the following are equivalent:*

*(i) The set $A$ is the set of maximal antichains of a partial order induced by (4) on $[M]$ for some $v_1 > \cdots > v_M > 0$.*

*(ii) The set $A$ is a set of mutually incomparable intervals that cover $[M]$.*

*Proof.* The fact that (i) implies (ii) is a simple calculation. Now assume (ii). For any interval $\{a, a+1, \ldots, b\} \in A$ we must have $\frac{v_c}{v_c + v_d} \le q$ for any $c, d \in \{a, a+1, \ldots, b\}$ for which $c < d$. From $v_a \ge v_c > v_d \ge v_b$ it follows that

$$\frac{v_a}{v_a + v_b} = \frac{1}{1 + \frac{v_b}{v_a}} \ge \frac{1}{1 + \frac{v_d}{v_c}} = \frac{v_c}{v_c + v_d}.$$

Thus, it suffices to show that there are real numbers $v_1 > \cdots > v_n > 0$ such that $\frac{v_a}{v_a + v_b} \le q$ for any $\{a, a+1, \ldots, b\} \in A$ and $\frac{v_c}{v_c + v_d} > q$ for any $c < d$ which are not contained in an antichain from $A$. We proceed by induction on $M$.

The induction base $M = 1$ is trivial. Assume $M \ge 2$. Since all elements of $A$ are intervals and any two intervals are mutually incomparable, it follows that $M$ is contained in exactly one set from $A$—possibly the singleton $\{M\}$. Let $A'$ be the set $A$ without the unique interval $\{a, a+1, \ldots, M\}$ containing $M$. Then $A'$ is a set of intervals that cover a proper subset $[M']$ of $[M]$ and fulfill the assumptions of (ii) for $[M']$. Hence by induction there is a choice of real numbers $v_1 > \cdots > v_{M'} > 0$ such that the set of maximal antichains of the order on $[M']$ induced by (4) is exactly $A'$. We consider two cases: (i) $a = M' + 1$. Then, by the considerations above, we need to choose numbers $v_{M'} > v_a > v_{a+1} > \cdots > v_M > 0$ such that $\frac{v_a}{v_a + v_M} \le q$ and $\frac{v_{M'}}{v_{M'} + v_a} > q$. The latter implies $\frac{d}{v_d + v_c} = \frac{1}{1 + \frac{v_c}{v_d}} \ge \frac{v_{M'}}{v_{M'} + v_a} > q$ for $d \le M' > a = M' + 1 \ge d \ge M$. But those are easily checked to exist. (ii) $a \le M'$. Since $M'$ is contained in at least one set from $A'$ and since this set is not contained in $\{a, a+1, \ldots, M\}$, it follows that $q \ge \frac{v_{a-1}}{v_{a-1} + v_{M'}} > \frac{v_a}{v_a + v_{M'}}$. In particular $(1 - q)v_a < qv_{M'}$. Now choose $v_{M'+1} > v_{M'+2} > \cdots > v_M > 0$ such that $qv_{M'} > qv_{M'+1} > qv_M > v_a(1 - q)$. Note that here $q > 1/2$ is essential. Then one checks that all desired inequalities are fulfilled. $\square$

**Theorem 4.** *Let $\mathcal{Q}_{PL}$ denote the set of different partial orders (up to isomorphism) that can be represented as a threshold relation $Q$ defined by (1), where $P$ is derived according to (4) from the PL model (3). For any given threshold $q \in [1/2, 1)$, the cardinality of this set is given by the $M^{th}$ Catalan number:*

$$|\mathcal{Q}_{PL}| = \frac{1}{M+1}\binom{2M}{M}$$

*Sketch of Proof.* Without loss of generality, we can assume the parameters of the PL model to satisfy

$$v_1 > \cdots > v_M > 0 \ . \tag{5}$$

Consider the $(M \times M)$-matrix with entries $P(i, j)$. By (5), the main diagonal of this matrix is strictly increasing along rows and columns. From the set $\{(i, j) \mid 0 \leq i \leq M+1, 0 \leq i-1 \leq j \leq M\}$, we remove those $(i, j)$, $1 \leq i < j \leq M$, for which $P(i, j)$ is above the given threshold. As a picture in the plane, this yields a shape whose upper right boundary can be identified with a Dyck path—a path on integer points consisting of $2M$ moves $(1, 0)$, $(0, 1)$ from position $(1, 0)$ to $(M+1, M)$ and staying weakly above the $(i+1, i)$-diagonal. It is immediate that each path uniquely determines its partial order. Moreover, it is well-known that these Dyck paths are counted by the $M^{th}$ Catalan number.

In order to verify that any Dyck path is induced by a suitable choice of parameters, one establishes a bijection between Dyck paths from $(1, 0)$ to $(M+1, M)$ and maximal sets of mutually incomparable intervals (in the natural order) in $[M]$. To this end, consider for a Dyck path a peak at position $(i, j)$, i.e., a point on the path where a $(1, 0)$ move is followed by a $(0, 1)$ move. Then we must have $j \geq i$, and we identify this peak with the interval $\{i, i+1, \ldots, j\}$. It is a simple yet tedious task to check that assigning to a Dyck path the set of intervals associated to its peaks is indeed a bijection to the set of maximal sets of mutually incomparable intervals in $[M]$. Again, it is easy to verify that the set of intervals associated to a Dyck path is the set of maximal antichains of the partial order determined by the Dyck path. Now, the assertion follows from Lemma 5.

Again, using Lemma 5, one checks that (5) implies that partial orders induced by (4) in the PL model have a unique labeling up to poset automorphism. Hence our count is a count of isomorphism classes. $\square$

We note that, from the above proof, it follows that the partial orders in $\mathcal{Q}_{PL}$ are the so called *semiorders*. We refer the reader to Ch. 8 §2 [26] for more details. Indeed, the first part of the proof of Theorem 4 resembles the proof of Ch. 8 (2.11) [26]. Moreover, we remark that $\mathcal{Q}_M$ is not only smaller in size than $\mathcal{Q}_{PL}$, but the former is indeed strictly included in the latter: $\mathcal{Q}_M \subset \mathcal{Q}_{PL}$. This can easily be seen by defining the weights $v_i$ of the PL model as $v_i = 2^{M-i}$ ($i \in [M]$), in which case the matrix with entries $P(i, j) = \frac{2^{j-i}}{1+2^{j-i}}$ is Toeplitz.

Finally, given that we have been able to derive explicit (combinatorial) expressions for $|\mathcal{Q}_M|$ and $|\mathcal{Q}_{PL}|$, it might be interesting to note that, somewhat surprisingly at first sight, no such expression exists for the *total* number of partial orders on $M$ elements.

## 5 Experiments

We complement our theoretical results by an empirical study, in which we compare the different approaches on the SUSHI data set,[1] a standard benchmark for preference learning. Based on a food-chain survey, this data set contains complete rankings of 10 types of sushi provided by 5000 customers, where each customer is characterized by 11 numeric features.

Our evaluation is done by measuring the tradeoff between *correctness* and *completeness* achieved by varying the threshold $q$ in (1). More concretely, we use the measures that were proposed in [6]: correctness is measured by the *gamma rank correlation* between the true ranking and the predicted partial order (with values in $[-1, +1]$), and completeness is defined by one minus the fraction of pairwise comparisons on which the model abstains. Obviously, the two criteria are conflicting: increasing completeness typically comes along with reducing correctness and vice versa, at least if the learner is effective in the sense of abstaining from those decisions that are indeed most uncertain.

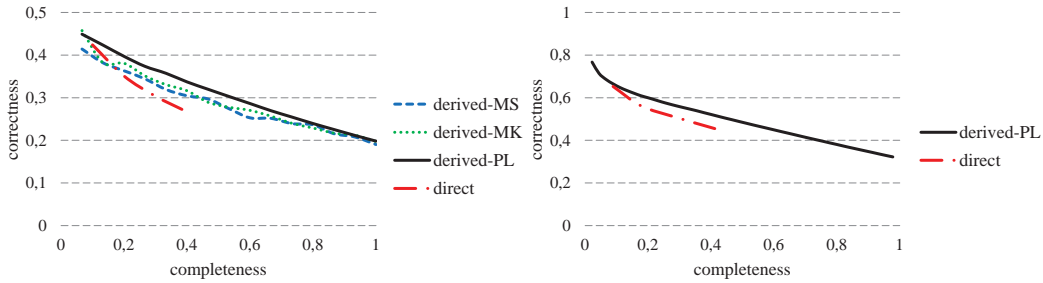

Figure 1: Tradeoff between completeness and correctness for the SUSHI label ranking data set: Existing pairwise method (direct) versus the probabilistic approach based on the PL model and Mallows model with Spearman's rho (MS) and Kendall (MK) as distance measure. The figure on the right corresponds to the original data set with rankings of size 10, while and the figure on the left shows results for rankings of size 6.

We compare the original method of [6] with our new proposal, calling the former *direct*, because the pairwise preference degrees on which we threshold are estimated directly, and the latter *derived*, because these degrees are derived from probability distributions on $\Omega$. As a label ranker, we used the instance-based approach of [5] with a neighborhood size of 50. We conducted a 10-fold cross validation and averaged the completeness/correctness curves over all test instances. Due to computational reasons, we restricted the experiments with the Mallows model to a reduced data set with only six labels, namely the first six of the ten sushis. (The aforementioned label ranker is based on an instance-wise maximum likelihood estimation of the probability distribution $\mathbf{P}$ on $\Omega$; in the case of the Mallows model, this involves the estimation of the center ranking $\pi_0$, which is done by searching the discrete space $\Omega$, that is, a space of size $|M!|$.)

The experimental results are summarized in Figure 1. The main conclusion that can be drawn from these results is that, as expected, our probabilistic approach does indeed achieve a better tradeoff between completeness and correctness than the original one, especially in the sense that it spans a wider range of values for the former. Indeed, with the direct approach, it is not possible to go beyond a completeness of around 0.4, whereas our probabilistic methods always allow for predicting complete rankings (i.e., to achieve a completeness of 1). Besides, we observe that the tradeoff curves of our new methods are even lifted toward a higher level of correctness. Among the probabilistic models, the PL model performs particularly well, although the differences are rather small.

Similar results are obtained on a number of other benchmark data sets for label ranking. These results can be found in the supplementary material.

## 6   Summary and Conclusions

The idea of producing predictions in the form of a partial order by thresholding a (valued) pairwise preference relation is meaningful in the sense that a learner abstains on the most unreliable comparisons. While this idea has first been realized in [6] in an ad-hoc manner, we put it on a firm mathematical grounding that guarantees consistency and, via variation of the threshold, allows for exploiting the whole spectrum between a complete ranking and an empty relation.

Both variants of our probabilistic approach, the one based on the Mallows and the other one based the PL model, are theoretically sound, semantically meaningful, and show strong performance in first experimental studies. The PL model may appear especially appealing due to its expressivity, and is also advantageous from a computational perspective.

An interesting question to be addressed in future work concerns the possibility of improving this model further, namely by increasing its expressivity while still assuring consistency. In fact, the transitivity properties guaranteed by PL seem to be stronger than what is necessarily needed. In this regard, we plan to study models based on the notion of Luce-decomposability [20], which include PL as a special case.

## Footnotes

[1]Available online at `http://www.kamishima.net/sushi`

# References

[1] P.L. Bartlett and M.H. Wegkamp. Classification with a reject option using a hinge loss. *Journal of Machine Learning Research*, 9:1823–1840, 2008.

[2] E.V. Bonilla, S. Guo, and S. Sanner. Gaussian process preference elicitation. In *Proc. NIPS–2010*, pages 262–270, Vancouver, Canada, 2010. MIT Press.

[3] R. Bradley and M. Terry. Rank analysis of incomplete block designs. I: The method of paired comparisons. *Biometrika*, 39:324–345, 1952.

[4] W. Cheng, K. Dembczyński, and E. Hüllermeier. Label ranking based on the Plackett-Luce model. In *Proc. ICML–2010*, pages 215–222, Haifa, Israel, 2010. Omnipress.

[5] W. Cheng, J. Hühn, and E. Hüllermeier. Decision tree and instance-based learning for label ranking. In *Proc. ICML–2009*, pages 161–168, Montreal, Canada, 2009. Omnipress.

[6] W. Cheng, M. Rademaker, B. De Baets, and E. Hüllermeier. Predicting partial orders: Ranking with abstention. In *Proc. ECML/PKDD–2010*, pages 215–230, Barcelona, Spain, 2010. Springer.

[7] C. Chow. On optimum recognition error and reject tradeoff. *IEEE Transactions on Information Theory*, 16(1):41–46, 1970.

[8] W. Chu and Z. Ghahramani. Preference learning with Gaussian processes. In *Proc. ICML–2005*, pages 137–144, Bonn, Germany, 2005. ACM.

[9] D. Critchlow, M. Fligner, and J. Verducci. Probability models on rankings. *Journal of Mathematical Psychology*, 35:294–318, 1991.

[10] O. Dekel, CD. Manning, and Y. Singer. Log-linear models for label ranking. In *Proc. NIPS–2003*, Vancouver, Canada, 2003. MIT Press.

[11] P. Diaconis. *Group representations in probability and statistics*, volume 11 of *Lecture Notes–Monograph Series*. Institute of Mathematical Statistics, Hayward, CA, 1988.

[12] P.C. Fishburn. Binary choice probabilities: on the varieties of stochastic transitivity. *Journal of Mathematical Psychology*, 10:321–352, 1973.

[13] A. Gionis, H. Mannila, K. Puolamäki, and A. Ukkonen. Algorithms for discovering bucket orders from data. In *Proc. KDD–2006*, pages 561–566, Philadelphia, US, 2006. ACM.

[14] I.C. Gormley and T.B. Murphy. A latent space model for rank data. In *Proc. ICML–06*, pages 90–102, Pittsburgh, USA, 2006. Springer.

[15] J. Guiver and E. Snelson. Bayesian inference for Plackett-Luce ranking models. In *Proc. ICML–2009*, pages 377–384, Montreal, Canada, 2009. Omnipress.

[16] S. Har-Peled, D. Roth, and D. Zimak. Constraint classification: a new approach to multiclass classification. In *Proc. ALT–2002*, pages 365–379, Lübeck, Germany, 2002. Springer.

[17] G. Lebanon and Y. Mao. Nonparametric modeling of partially ranked data. *Journal of Machine Learning Research*, 9:2401–2429, 2008.

[18] T. Lu and C. Boutilier. Learning Mallows models with pairwise preferences. In *Proc. ICML–2011*, pages 145–152, Bellevue, USA, 2011. Omnipress.

[19] R. Luce and P. Suppes. *Handbook of Mathematical Psychology*, chapter Preference, Utility and Subjective Probability, pages 249–410. Wiley, 1965.

[20] J. Marden. *Analyzing and Modeling Rank Data*. Chapman and Hall, 1995.

[21] M. Meila and H. Chen. Dirichlet process mixtures of generalized mallows models. In *Proc. UAI–2010*, pages 358–367, Catalina Island, USA, 2010. AUAI Press.

[22] T. Qin, X. Geng, and T.Y. Liu. A new probabilistic model for rank aggregation. In *Proc. NIPS–2010*, pages 1948–1956, Vancouver, Canada, 2010. Curran Associates.

[23] M. Rademaker and B. De Baets. A threshold for majority in the context of aggregating partial order relations. In *Proc. WCCI–2010*, pages 1–4, Barcelona, Spain, 2010. IEEE.

[24] R.P. Stanley. *Enumerative Combinatorics, Vol. 2*. Cambridge University Press, 1999.

[25] L. Thurstone. A law of comparative judgment. *Psychological Review*, 79:281–299, 1927.

[26] W.T. Trotter. *Combinatorics and partially ordered sets: dimension theory*. The Johns Hopkins University Press, 1992.

